# United We Stand, Divided We Fall: Fingerprinting Deep Neural Networks via Adversarial Trajectories

**Tianlong Xu[1], Chen Wang[1], Gaoyang Liu[\*1], Yang Yang[2], Kai Peng[1], Wei Liu[1]**
[1]Hubei Key Laboratoryof Smart Internet Technology, School of EIC,
Huazhong University of Science and Technology, [2]School of AI, Hubei University
[1]{tianlongxu, chenwang, liugaoyang, pkhust, liuwei}@hust.edu.cn, [2]yangyang@hubu.edu.cn

## Abstract

In recent years, deep neural networks (DNNs) have witnessed extensive applications, and protecting their intellectual property (IP) is thus crucial. As a non-invasive way for model IP protection, model fingerprinting has become popular. However, existing single-point based fingerprinting methods are highly sensitive to the changes in the decision boundary, and may suffer from the misjudgment of the resemblance of sparse fingerprinting, yielding high false positives of innocent models. In this paper, we propose ADV-TRA, a more robust fingerprinting scheme that utilizes adversarial trajectories to verify the ownership of DNN models. Benefited from the intrinsic progressively adversarial level, the trajectory is capable of tolerating greater degree of alteration in decision boundaries. We further design novel schemes to generate a surface trajectory that involves a series of fixed-length trajectories with dynamically adjusted step sizes. Such a design enables a more unique and reliable fingerprinting with relatively low querying costs. Experiments on three datasets against four types of removal attacks show that ADV-TRA exhibits superior performance in distinguishing between infringing and innocent models, outperforming the state-of-the-art comparisons.

## 1 Introduction

In recent years, deep neural networks (DNNs) have witnessed extensive applications, such as autonomous driving [1], AIGC [2] and medical diagnosis [3]. However, training a practical DNN model requires significant computational resources, data and time. For example, the training of GPT-3 took about 21 days with a cost of over $2.4$ million dollars [4]. It is thus essential to protect the intellectual property (IP) of DNN models, especially when the models face the risk of being exposed or stolen by the so-called model extraction attacks [5].

Current technologies to protect model IP can be broadly categorized into two classes: model watermarking [6, 7, 8] and model fingerprinting [9, 10, 11, 12]. While the former embeds the watermark into the model to verify the identity of suspect models, it inevitably interferes in the training phase, which sacrifices the utility of the model or even introduces new security threats [7, 13]. So the latest research trend has shifted toward model fingerprinting, which enables the extraction of a model's fingerprint without any modifications to the model itself.

A common practice of model fingerprinting methods is to generate a batch of special adversarial samples near the decision boundary as fingerprinting samples for model verification [14, 12]. However, these *single-point* fingerprinting samples are generated independently from each other, and are highly sensitive to the changes in the decision boundary, given their

---

[\*]Corresponding Author

inherent localized perspective. When the decision boundary changes, e.g., due to removal attacks [15, 16], a great number of such fingerprinting samples would become invalid (c.f. Figure 1a).

Moreover, even two unrelated models may share similar portions of decision boundaries (i.e., the resemblance of sparse fingerprinting) [17, 18]. Therefore, single-point fingerprints are more prone to incorrectly identifying an unrelated (innocent) model as stolen, yielding a high false positive rate in the verification process, as observed in [10, 12].

In this paper, we propose ADV-TRA, a more robust fingerprinting method for DNN models. Instead of using single-point fingerprinting samples to identify the model, ADV-TRA exploits novel adversarial trajectories, each of which is a chain of progressively adversarial samples[2], representing adversarial level from weak to strong (c.f. Figure 1b). The adversarial trajectory, incorporating multi-level adversarial perturbations,

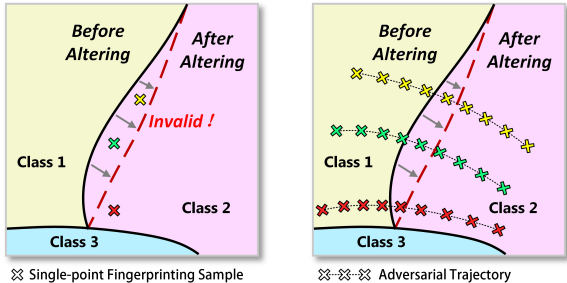

(a) Single-point Fingerprinting    (b) Trajectory Fingerprinting

Figure 1: Comparing single-point fingerprinting samples and our adversarial trajectories. As can be seen, when the decision boundary alters (from the solid black line to the dashed red line), (a) a portion of single-point fingerprinting samples near the boundary become invalid, while (b) the majority of samples in the trajectories remain effective.

could provide a more precise localization of the decision boundary, which enables to accurately capture the alteration in the boundary induced by removal attacks, thus exhibiting a more robust model IP verification.

Though the idea is simple, there are two major challenges. First, it is difficult to control the trajectory precisely to pass through the decision boundary with fixed length (i.e., the number of samples in the trajectory). Fixed step sizes in existing adversarial sample generation may result in too long or too short trajectories, yielding substantial querying times and unstable local optima, or compromising effectiveness. To tackle this issue, we view each step size as an optimized variable and design several loss functions and strategies to enforce fixed-length trajectories with dynamically adjusted step sizes. This leads to smaller steps for samples near the boundary to focus on more subtle details, while larger steps for samples far away from the boundary to quickly move towards the boundary.

Second, it is not easy to capture the global fingerprint features to avoid the misjudgment of the resemblance of sparse fingerprinting, since the trajectory across neighboring classes remains limited to features of a single decision boundary. To address this challenge, we propose to generate a surface trajectory that comprises numerous adversarial trajectories orderly across multiple classes, which can offer a more comprehensive representation of the decision surface (instead of a single boundary), thereby significantly reducing the false positives of innocent models.

Our major contributions can be summarized as follows:

- We propose ADV-TRA, a more robust fingerprinting scheme that utilizes adversarial trajectories for model verification. Benefited from the intrinsic progressively adversarial level, the trajectory is able tolerate greater degree of alteration in decision boundaries.

- We design novel schemes to generate a surface trajectory that involves a series of fixed-length trajectories with dynamically adjusted step sizes. Such a design enables a more unique and reliable fingerprinting with relatively low querying costs.

- We conduct extensive experiments on three datasets against four types of removal attacks. Experimental results show that ADV-TRA exhibits superior performance in distinguishing between infringing and innocent models, outperforming the state-of-the-art comparisons.

## 2 Preliminary

### 2.1 Adversarial Samples

Given a target model, the goal of adversarial samples is to deceive the model into making incorrect predictions [19, 20]. In the context of a classification model represented as $f$, an adversarial sample, $\hat{x} = x + \delta$, is crafted to perturb a clean sample $x$ with a ground-truth label $y$ in such a way that: (1) the perturbation $\delta$ is kept small; and (2) the predicted class for $\hat{x}$ is altered. This can be formalized as:

$$\min \ \|x - \hat{x}\| \ \text{ s.t. } \ f(\hat{x}) \neq y, \tag{1}$$

where $\|\cdot\|$ is a distance metric, e.g., $\ell_2$ distance. In this way, the clean sample $x$ and its perturbed counterpart $\hat{x}$ may appear nearly identical to human observers, but the target model perceives them as entirely distinct entities.

### 2.2 Model Fingerprinting

As a non-invasive way to validate the ownership of a DNN model, model fingerprinting seeks to detect some fingerprints which is normally the model's decision boundary. Generally, the procedure of model fingerprinting involves two main steps: fingerprint extraction and fingerprint verification:

**Fingerprint Extraction.** Suppose a model owner who trains a source model, his goal is to generate a number of fingerprinting samples that can uniquely characterize the decision boundaries. To this end, model fingerprinting borrows ideas from adversarial samples, by generating a set of adversarial examples that are close to the decision boundary, paired with their corresponding predicted labels, as the fingerprinting samples [9, 11, 14]. More specifically, given a normal sample $x$ and its ground truth label $y$, the fingerprinting sample $x_{fp}$ can be derived from optimizing towards $y_{fp}$, where $y_{fp}$ is the fingerprinting label that differs from $y$. Therefore, for a source model $f_{src}$, the corresponding fingerprinting sample $x_{fp}$ just crosses the decision boundaries, such that $f_{src}(x_{fp}) \neq f_{src}(x)$.

**Fingerprint Verification.** During the verification phase, when dealing with a suspect model $f_{sus}$, the model owner can determine the ownership by querying it with the set of fingerprinting samples $D_{fp} = \{(x_{fp}, y_{fp})\}$. Based on the output results, the owner selects a testing metric $Metric(\cdot)$, e.g., the accuracy, and computes $Metric(f_{sus}, D_{fp})$. By comparing this result with a threshold, the owner can make the final judgment regarding any potential ownership infringement.

### 2.3 Removal Attacks

Removal attacks are designed to invalidate fingerprints by tampering with the source model $f_{src}$. We denote a model that has been processed by removal attacks as $f_{rmv}$, with the goal being to alter the model such that $f_{rmv}(x_{fp}) \neq f_{src}(x_{fp})$. There are several ways to launch removal attacks, e.g., by model modification (fine-tuning [21], pruning [22, 23], adversarial training [24, 25]), or model extraction [26, 5]. Removal attacks pose a significant threat to model fingerprinting. For example, when the source model undergoes adversarial training, a considerable portion of these fingerprints may change (i.e., fingerprinting samples become obsolete in the verification phase) [9, 12, 16].

## 3 Problem Formation

We consider a typical scenario consisting of two parties: the model owner (defender) and the attacker, as shown in Figure 2. The model owner trains a well-performed model (i.e., the source model) and deploys it as a cloud service or client-sided software. The attacker attempts to steal the source model in either black-box or white-box ways.

**Attacker's Strategy.** We first consider a strong white-box attacker, who can get access to the entire information of the source model, including the model structure and all inner parameters, as well as

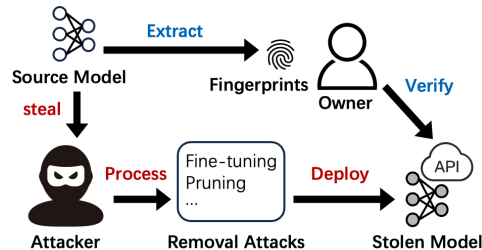

Figure 2: Illustration of model stealing and verification.

sufficient auxiliary data[3], through server intrusion or
eavesdropping on communication channels. Once the attacker acquires the source model, he could
leverage removal attacks to modify the model from either the model structure or parameters, so as to
evade the IP infringement detection.

We also consider a black-box attacker, who only has the input-output prediction interface (a.k.a. API)
to the source model. By leveraging an auxiliary dataset, the attacker is able to duplicate the source
model by multiple querying, e.g., model extraction attack, and utilizing the querying results to train a
substitute model from scratch.

**Defender's Strategy.** The goal of the model owner is to verify whether a deployed suspect model is
derived from the source model. The model owner naturally has the white-box access to the source
model, but is assumed to only have the black-box access to the suspect model[4] and observe its
prediction for any input. In specific, he can extract a set of fingerprinting samples from the source
model and assess their performance on the suspect model. If the number of fingerprinting samples
that validate successfully exceeds a threshold, then it can be concluded that the suspect model has
been illegally stolen from the source model.

From the perspective of the defender, we aim to design a DNN fingerprinting method satisfying the
following properties: (1) **Uniqueness.** Uniqueness is crucial for ensuring the reliability and fidelity
of a fingerprinting scheme. Fingerprint extraction should comprehensively and fully capture features,
so as to avoid local resemblance-induced misjudgment. (2) **Robustness.** A robust fingerprinting
scheme needs to be resistant to removal attacks, even under relatively radical attacking strategies. (3)
**Efficiency.** Considering the API may charge per query and excessive queries may raise the attacker's
suspicion, the fingerprinting method may require as few query samples as possible.

## 4    Design of ADV-TRA

We present the workflow of ADV-TRA in Figure 3. In the trajectory generation phase, given the
source model $f_{src}$ and a base sample $x_0$ of class $c_0$, ADV-TRA first initializes the trajectory from one
class $c_0$ towards another class $c_1$. Subsequently, the trajectory is fine-tuned for probing the decision
boundary and then simply bilateralized to enforce a cross-class chain of progressively adversarial
samples. Finally, we introduce the surface trajectory $\mathcal{T}_{surf}$ which traverses among multiple classes to
fingerprint the entire decision surface. In the verification phase, we use $\mathcal{T}_{surf}$ to compute the mutation
rate $r_{mut}$ for a suspect model $f_{sus}$, and determine whether $f_{sus}$ is stolen from $f_{src}$.

### 4.1    Trajectory Generation

We use adversarial trajectories to fingerprint the source model's decision boundaries. A naive way of
generating the adversarial trajectory is to record all intermediate products using adversarial sample
algorithms like Basic Iterative Method (BIM) [29]. However, this approach does not consider the
distance from a base sample $x_0$ to its corresponding decision boundary, and often uses a fixed step
size, which would cause too few (or too many) trajectory samples, resulting in lower effectiveness (or
excessive queries, even getting stuck in local optima and unable to cross the decision boundary) .

Therefore, we propose an adaptive step-size adversarial trajectory generation algorithm that can
achieve fixed length between any two classes by dynamically adjusting the step size, by involving the
following four steps: (1) trajectory initialization; (2) boundary probing; (3) trajectory bilateralization;
and (4) trajectory connection.

**Trajectory Initialization.** Given a source model $f_{src}$, a base sample $x_0$ (with its ground truth label
$y_{bs}$), and a target label $y_{tgt}$, a trajectory is initialized just like the process of generating adversarial
samples with gradient descent. Instead of using fixed step size, we view the step size $s_i$ in all rounds
of iterations $\mathbf{s} = \{s_0, s_1, \ldots, s_{l-1}\}$ as an optimizable variable and generate the *initial trajectory* $\mathcal{T}_{int}$:

main [26] or just synthesized using image generation techniques [27, 28].

[4]We consider the most difficult and practical settings where the attacker deploys the stolen model on the
cloud and only the prediction interface is provided for user's queries. Our method can be easily adapted to
white-box cases where richer information such as the outputs of each layer of the model can be obtained.

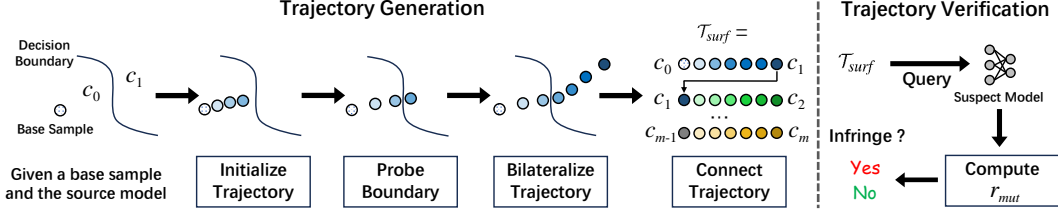

Figure 3: Exemplary illustration of ADV-TRA pipeline (the length of $\mathcal{T}_{int}$ $l = 3$; the length of $\mathcal{T}_{surf}$ is $2lm$, exclusive of $c_0$). A darker color of the trajectory sample indicates a higher level of adversarial strength towards the target class.

$$\mathcal{T}_{int} = \mathcal{T}(f_{src}, x_0, y_{tgt}, \mathbf{s}) = \{x_0, x_1, \ldots, .x_l\}$$
$$\text{s.t. } x_{i+1} = x_i - s_i \cdot \text{sign}(\nabla\mathcal{L}(f_{src}, x_i, y_{tgt})), \tag{2}$$
$$\text{for } i = 0, 1, \ldots, l - 1,$$

where $\mathcal{L}(f_{src}, x_i, y_{tgt})$ is the loss function to guide the prediction towards $y_{tgt}$, and $l$ is the length of $\mathcal{T}_{int}$.

**Boundary Probing.** Recall that we use $\mathcal{T}_{int}$ of length $l$ to probe the decision boundary, so it is crucial to scale $\mathcal{T}_{int}$ such that it precisely reaches the decision boundary, i.e., the last sample $x_l$ in $\mathcal{T}_{int}$ happens to predicted as $y_{tgt}$ (whereas the prior samples are not):

$$f_{src}(x_i) \neq y_{tgt}, \quad \text{if } i = 1, 2, \ldots, l - 1$$
$$f_{src}(x_i) = y_{tgt}, \quad \text{if } i = l. \tag{3}$$

To this end, we propose a simple length control strategy, which scales $\mathcal{T}_{int}$ by adjusting the step sizes $\mathbf{s}$ with a length control variable $\alpha_{lc}$ ($0 < \alpha_{lc} < 1$). Specifically, given a trajectory, we first check whether the requirements in Eq. (3) are satisfied, and further determine whether to increase or decrease the step size. If the trajectory reaches the decision boundary early (late), we need to reduce (resp. enlarge) the step size of all steps $\mathbf{s}$ by multiplying a length control factor $\alpha_{lc}$ (resp. $1/\alpha_{lc}$).

In order to record more details near the decision boundary, the step size $s_i$ is expected to decay when current $x_i$ are closer to the decision boundary, i.e., $s_{i+1} < s_i$. For this purpose, we define the brake loss function $\mathcal{L}_{brk}$ as:

$$\mathcal{L}_{brk}(\mathbf{s}) = \sum_{i=0}^{l-1}(\alpha_{brk} \cdot s_i - s_{i+1})^2, \tag{4}$$

where $\alpha_{brk}$ ($0 < \alpha_{brk} < 1$) is the brake factor that controls the proportional relationship of two adjacent step sizes.

Moreover, to ensure the internal forward trend within the trajectory, i.e., $x_{i+1}$ is closer to the decision boundary of $y_{tgt}$ than $x_i$, each step $s_i$ should be greater than 0. So we define the forward loss function $\mathcal{L}_{fwd}$ to prevent any recession in the trajectory:

$$\mathcal{L}_{fwd}(\mathbf{s}) = \sum_{i=0}^{l-1}\mathbb{I}(s_i < 0) \cdot s_i, \tag{5}$$

where $\mathbb{I}(\cdot)$ is the indicator function.

Hence, training the step sizes $\mathbf{s}$ involves the above two optimization objectives:

$$Min : \mathcal{L} = \mathcal{L}_{brk}(\mathbf{s}) + \mathcal{L}_{fwd}(\mathbf{s}). \tag{6}$$

We optimize $\mathbf{s}$ using the gradient descent algorithm. Specifically, the optimization terminates when satisfying the following conditions:(1) The last sample $x_l$ in $\mathcal{T}_{int}$ is predicted as $y_{tgt}$, while the prior samples are not, as described in Eq. 3. (2) The step size for the next sample $s_{i+1}$ is smaller than the step size $s_i$ for the previous sample, i.e., $s_{i+1} < s_i$.

**Trajectory Bilateralization.** Given the optimized $\mathcal{T}_{int}$, we next extend it to a *bilateral trajectory* $\mathcal{T}_{bi}$. This is necessary due to the random variation in the perturb direction of the decision boundary

when subjected to removal attacks, and the trajectory is preferred to evenly attend to both sides of the decision boundary to keep a consistent measuring scale for both sides. We thus utilize a mirroring operation, centered on $x_l$, to make up the other half of the trajectory, using the reversed step sizes $\hat{\mathbf{s}} = \{s_{l-1}, s_{l-2}, \ldots, s_0\}$, and a bilateral trajectory $\mathcal{T}_{bi}$ can thus be generated as:

$$\mathcal{T}_{bi} = \mathcal{T}(f_{src}, x_0, y_{tgt}, \mathbf{s} \cup \hat{\mathbf{s}}) = \{x_0, x_1, \ldots, .x_{2l}\}, \tag{7}$$

where $\mathcal{T}_{bi}$ crosses to the other side of the decision boundary starting from $x_l$, and is able to locate the boundary well with the intrinsic progressively adversarial samples.

It is worth noting that the bilateral trajectory may pass through more than two classes. Such a case does not lower the efficacy of the trajectory; instead, it serves as a special feature regarding to the model's decision surface, which makes our fingerprinting more unique.

**Trajectory Connection.** So far we can fingerprint a specific decision boundary between two classes $c_0$ and $c_1$ by a bilateral trajectory $\mathcal{T}_{bi}^{c_0 \to c_1}$. However, a decison boundary alone, as discussed before, is not sufficient to fully capture the fingerprint of the model. While two unrelated models may share some similarities along the decision boundary, their decision surfaces would almost never be the same [17, 18]. Therefore, we introduce the *surface trajectory* $\mathcal{T}_{surf}$ to fingerprint the entire decision surface, which merges all the bilateral trajectories to traverse through all classes. This can be done by simply connecting the bilateral trajectories in order, i.e., the last sample in $\mathcal{T}_{bi}^{c_0 \to c_1}$ is the first sample of $\mathcal{T}_{bi}^{c_1 \to c_2}$ (c.f. Figure 3).

It is noticed that, it may be costly to involve all the bilateral trajectories, especially for complex models. Therefore, a more practical way for large models is to randomly select $m$ bilateral trajectories for the surface trajectory generation (c.f. Algorithm 1 in Appendix B):

$$\mathcal{T}_{surf} = \mathcal{T}_{bi}^{c_0 \to c_1} \cup \mathcal{T}_{bi}^{c_1 \to c_2} ... \cup \mathcal{T}_{bi}^{c_{m-1} \to c_m}, \tag{8}$$

Our experiments indicate that a small number of $m = 10$ is sufficient to fingerprint the 100-class CIFAR-100 and 1000-class ImageNet models.

## 4.2 Trajectory Verification

In the verification phase, the model owner can query the given suspect model $f_{sus}$ with the surface trajectory $\mathcal{T}_{surf}$. We then calculate the mutation rate of $\mathcal{T}_{surf}$ from $f_{src}$ to $f_{sus}$ by:

$$r_{mut} = \frac{1}{|\mathcal{T}_{surf}|} \sum_{x_i \in \mathcal{T}_{surf}} \mathbb{I}(f_{sus}(x_i) \neq f_{src}(x_i)). \tag{9}$$

Here, $r_{mut}$ reflects the proportion of samples in the trajectory whose predictions on $f_{sus}$ differ from $f_{src}$. Theoretically, an infringement model is expected to possess a lower $r_{mut}$ than an innocent model, since it originates from the source model $f_{src}$ and shares a more similar decision surface with it. The defender can determine whether $\mathcal{T}_{surf}$ is also the exclusive trajectory of $f_{sus}$ by comparing the obtained $r_{mut}$ and a threshold $Thr_{mut}$ (0.5 in our case, equivalent to random guess). In practice, we calculate the detection rate of a number of adversarial trajectories (i.e. surface trajectories) on the suspect model to make the final infringement judgment.

## 5 Experiments

We use three widely-used benchmark datasets, namely, CIFAR-10 [30], CIFAR-100 [30], and ImageNet [31] , in model fingerprinting domain to evaluate the performance of our ADV-TRA. We compare our approach with four existing model fingerprinting techniques, including **CoRt** [11], **IPGuard** [14], **CAE** [9], and **UAP** [10]. We also evaluate the robustness of ADV-TRA against four types of removal attacks (fine-tuning [6, 32], pruning [33], adversarial training [34], and model extraction attacks [26, 27, 35]) across multiple model architectures. In Appendix C we furthermore list details on experiment setups, including dataset, models, evaluation metrics, and implementation details.

Moreover, we reveal how removal attacks affect the effectiveness of fingerprints by altering the decision surface in Appendix D.1. Additionally, we compare a simple approach (trajectory spanning only two categories) with trajectories spanning multiple categories (see Appendix D.2), and conduct a series of ablation studies to analyze the impact of parameters of ADV-TRA (see Appendix D.3). At last, we demonstrate how the progressiveness of the trajectory enables it to locate the decision boundaries (see Appendix D.4).

Table 1: Main results on CIFAR-10 dataset. P-20% denotes model pruning with a pruning rate $p = 0.2$; Adv-0.001 represents adversarial training with budget $\epsilon = 0.001$. Fingerprint detection rate in bold indicates the best performance. For positive models, a higher fingerprint detection rate is preferred, suggesting better ability to verify IP infringement. In contrast, negative models are expected to yield a lower detection rate, avoiding false verification.

| Model Type | | CIFAR-10 | | Fingerprint Detection Rate | | | |
|---|---|---|---|---|---|---|---|
| | | Accuracy | $\bar{r}_{mut}$ | IPGuard | CAE | UAP | Ours |
| Source Model | | $0.874 \pm 0.000$ | $0.000 \pm 0.000$ | $1.000 \pm 0.000$ | $1.000 \pm 0.000$ | $1.000 \pm 0.000$ | $1.000 \pm 0.000$ |
| Positive Suspect Model | FTLL | $0.868 \pm 0.001$ | $0.043 \pm 0.002$ | $\mathbf{0.978 \pm 0.001}$ | $0.956 \pm 0.002$ | $0.958 \pm 0.001$ | $0.952 \pm 0.000$ |
| | FTAL | $0.867 \pm 0.001$ | $0.058 \pm 0.002$ | $0.942 \pm 0.002$ | $0.934 \pm 0.001$ | $\mathbf{0.954 \pm 0.002}$ | $0.950 \pm 0.000$ |
| | RTLL | $0.865 \pm 0.000$ | $0.030 \pm 0.001$ | $0.917 \pm 0.001$ | $0.921 \pm 0.001$ | $0.944 \pm 0.002$ | $\mathbf{0.947 \pm 0.001}$ |
| | RTAL | $0.864 \pm 0.000$ | $0.143 \pm 0.001$ | $0.621 \pm 0.001$ | $0.663 \pm 0.002$ | $0.719 \pm 0.001$ | $\mathbf{0.763 \pm 0.002}$ |
| | P-20% | $0.872 \pm 0.001$ | $0.035 \pm 0.001$ | $0.895 \pm 0.001$ | $0.924 \pm 0.000$ | $0.946 \pm 0.001$ | $\mathbf{0.955 \pm 0.000}$ |
| | P-40% | $0.869 \pm 0.000$ | $0.064 \pm 0.001$ | $0.783 \pm 0.001$ | $0.894 \pm 0.001$ | $0.934 \pm 0.001$ | $\mathbf{0.945 \pm 0.001}$ |
| | P-80% | $0.825 \pm 0.002$ | $0.257 \pm 0.003$ | $0.587 \pm 0.001$ | $0.628 \pm 0.001$ | $0.634 \pm 0.002$ | $\mathbf{0.678 \pm 0.001}$ |
| | Adv-0.001 | $0.859 \pm 0.001$ | $0.238 \pm 0.001$ | $0.423 \pm 0.000$ | $0.421 \pm 0.001$ | $0.513 \pm 0.002$ | $\mathbf{0.758 \pm 0.001}$ |
| | Adv-0.01 | $0.863 \pm 0.000$ | $0.394 \pm 0.001$ | $0.162 \pm 0.001$ | $0.158 \pm 0.001$ | $0.348 \pm 0.000$ | $\mathbf{0.612 \pm 0.001}$ |
| | Adv-0.1 | $0.865 \pm 0.001$ | $0.205 \pm 0.002$ | $0.184 \pm 0.001$ | $0.256 \pm 0.002$ | $0.402 \pm 0.002$ | $\mathbf{0.628 \pm 0.001}$ |
| Negative Suspect Model | SAST | $0.879 \pm 0.000$ | $0.714 \pm 0.000$ | $0.631 \pm 0.001$ | $0.561 \pm 0.001$ | $0.412 \pm 0.000$ | $\mathbf{0.143 \pm 0.000}$ |
| | SADT | $0.876 \pm 0.001$ | $0.722 \pm 0.001$ | $0.497 \pm 0.001$ | $0.422 \pm 0.002$ | $0.382 \pm 0.002$ | $\mathbf{0.126 \pm 0.000}$ |
| | DAST | $0.875 \pm 0.002$ | $0.762 \pm 0.001$ | $0.454 \pm 0.001$ | $0.389 \pm 0.002$ | $0.323 \pm 0.001$ | $\mathbf{0.124 \pm 0.001}$ |
| | DADT | $0.874 \pm 0.003$ | $0.787 \pm 0.001$ | $0.357 \pm 0.002$ | $0.219 \pm 0.001$ | $0.313 \pm 0.001$ | $\mathbf{0.116 \pm 0.001}$ |

## 5.1 Main Performance

We first validate ADV-TRA on 100 suspect models for each source model, including 50 *positive* models under various attacks and 50 *negative* models on CIFAR-10 dataset. In our paper, the positive models originate from the source model but have been processed by removal attacks, including model pruning, adversarial training, and fine-tuning (Fine-Tune Last Layer (**FTLL**), Fine-Tune All Layers (**FTAL**), Retrain Last Layer (**RTLL**), and Retrain All Layers (**RTAL**)). The negative models come from four ways: (1) **SAST**: the same architecture as victim model and the same training data (Note that model initialization and training randomness can still lead to differences from the source model); (2) **SADT**: the same architecture as victim model but different training data; (3) **DAST**: a different architecture but the same training data; (4) **DADT**: a different architecture and different training data (For more details, please refer to Appendix C). The results are shown in Table 1.

It is evident that our ADV-TRA outperforms other methods in most cases (12 out of 14), achieving high detection rates for positive models while maintaining low detection rates for negative models. Only ADV-TRA is able to clearly separate the two types of models, which is adequate to guarantee the subsequent verification without any false positives. Its corresponding lower bound of positive models is $0.612$ (corresponds to Adv-0.01), which is still much higher than the upper bound of negative models ($0.143$, corresponds to SAST). In contrast, other methods fail to distinguish between some of the positive models and negative models. It is also surprising to see that our method has a low false positive rate: the detection rate of ADV-TRA for negative models is much lower than the three baselines. This is attributed to our design of the chain of progressively adversarial samples, which is able to simultaneously capture more characteristics at various levels to avoid false positives.

One intermediate product average mutation rate $\bar{r}_{mut}$ reflects the dissimilarity in the decision boundary of a source model and a suspect model. It can be observed that a removal attack causing higher $\bar{r}_{mut}$ brings about lower fingerprint detection rate. For example, three fine-tuning methods (**FTLL**, **FTAL**, and **RTLL**) only result in a $\bar{r}_{mut}$ of less than $0.06$, corresponding to a relatively high fingerprint detection rate of approximately $0.95$. Negative models have a high $\bar{r}_{mut}$ (over $0.7$) since their decision boundaries differ more significantly from the source model.

To further validate the ability of the fingerprinting samples to accurately distinguish between positive and negative models, we present the ROC curve and the distribution of all suspect models in Figure 4a and Figure 4b. Only ADV-TRA gets an $AUC = 1$, significantly outperforming other approaches. As can be seen, CoRt, IPGuard, and CAE all do not readily distinguish between the two types of suspect models, exhibiting strong cross-model transferability. Especially for general adversarial samples (CoRt), their transferability to negative models is more obvious, leading to high detection rate. For ADV-TRA, the detection rate of the negative model is quite low, approximately $0.15$. This also serves as empirical evidence for our analysis that our trajectories have an extremely low false positive rate.

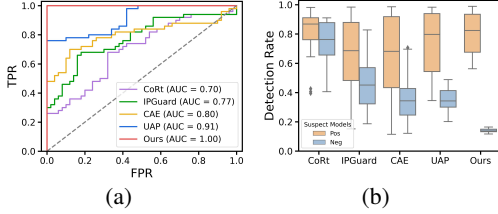
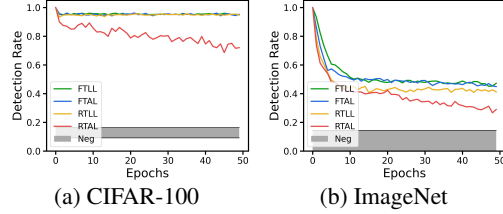

| (a) | (b) | (a) CIFAR-100 | (b) ImageNet |

Figure 4: The ROC curve (a) and the distribution (b) of fingerprint detection rate of different suspect models on CIFAR-10.

Figure 5: Fingerprint detection rate vs. fine-tuning epochs. The gray area represents the range of fingerprint detection rate for negative models.

Table 2: Comparison of different fingerprinting techniques. T1%F denotes True Positive Rate when the False Positive Rate is at 1%.

| Method | CIFAR-10 | | | CIFAR-100 | | | ImageNet | | |
|--------|-------|--------|------|-------|--------|------|-------|--------|------|
| | T1%F | T10%F | AUC | T1%F | T10%F | AUC | T1%F | T10%F | AUC |
| CoRt | 0.26 | 0.32 | 0.70 | 0.29 | 0.38 | 0.72 | 0.20 | 0.28 | 0.59 |
| IPGuard | 0.30 | 0.46 | 0.77 | 0.43 | 0.63 | 0.80 | 0.27 | 0.40 | 0.69 |
| CAE | 0.48 | 0.64 | 0.80 | 0.51 | 0.70 | 0.83 | 0.39 | 0.48 | 0.76 |
| UAP | 0.76 | 0.78 | 0.91 | 0.77 | 0.83 | 0.92 | 0.46 | 0.56 | 0.85 |
| **Ours** | **1.00** | **1.00** | **1.00** | **1.00** | **1.00** | **1.00** | **0.88** | **0.90** | **0.96** |

To evaluate the universality of our method, we conduct experiments on three datasets as illustrated in Table 2. It can be found that our method extracts the most strongly detectable fingerprints, which achieves AUC of $1.0$ on CIFAR-10 and CIFAR-100. In all cases, ADV-TRA has better AUC, T1%F, and T10%F than other fingerprinting methods. For the models trained on ImageNet, we conjecture that classification models with more classes have more complex decision boundaries. When subjected to removal attacks, the model's functionality (such as predicting the probability of each class) is more prone to changes, resulting in less robustness compared to models with fewer classes.

## 5.2 Robustness Against Removal Attacks

**Impact of Fine-tuning.** We first employ a more radical fine-tuning strategy (i.e., with a higher learning rate and many more epochs) to better simulate real-world removal attacks. Figure 5 shows the results on the CIFAR-100 and ImageNet datasets. For CIFAR-100 dataset, we can see that the FTLL, FTAL, and RTLL models are hardly affected and consistently demonstrate a high fingerprint detection rate (around $0.96$), whereas the detection rate for the RTAL model steadily decreases, dropping to $0.72$ after 50 epochs. As for ImageNet dataset, which has more classes, the detection rate of the positive models experiences a more significant decline, dropping from $1.0$ to below $0.5$ within the first 12 fine-tuning epochs. After that, the FTLL, FTAL, RTLL models begin to stabilize, but the RTAL model continues declining at a high rate. When the epoch reaches 50, the detection rate of the RTAL model is $0.290$, still significantly higher than the upper bound of the negative models ($0.143$). In general, fine-tuning is not a considerable threat to ADV-TRA.

**Impact of Pruning.** Next, we test the impact of pruning, by varying the pruning rate from $10\%$ to $90\%$. Figure 6 illustrates the fingerprint detection rate and model accuracy for each pruning level on CIFAR-100 and ImageNet. We can clearly see that the detection rate has a strong correlation with the model accuracy. For CIFAR-100 dataset, as the pruning rate increases from $10\%$ to $60\%$, the fingerprint detection rate decreases slightly from $0.986$ to $0.920$. When the pruning rate exceeds $70\%$, the fingerprint detection rate experiences a sharp decline (from $0.832$ to $0.367$). The results on ImageNet dataset also reveal two declining lines with increasing curvature. Even at a pruning rate of $90\%$, the fingerprinting samples could no longer reliably differentiate between positive ($0.114 \pm 0.064$) and negative ($0.001 \sim 0.143$) models. However, when so many fingerprinting samples become invalid, the model accuracy also suffers a sharp decrease. In practice, considering the model's basic functionality, an attacker would not implement pruning attack of such strength.

**Impact of Adversarial Training.** To investigate how adversarial training affects the effectiveness of fingerprinting samples, we vary the perturbation budget $\epsilon$ from $0.001$ to $1.0$ and attack the source

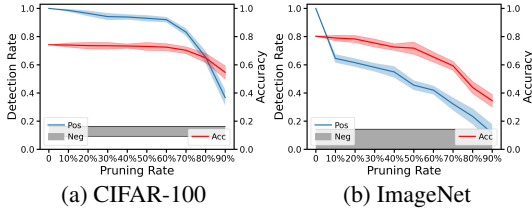

(a) CIFAR-100    (b) ImageNet

Figure 6: Fingerprint detection rate (blue line) and model accuracy (red line) under pruning attacks.

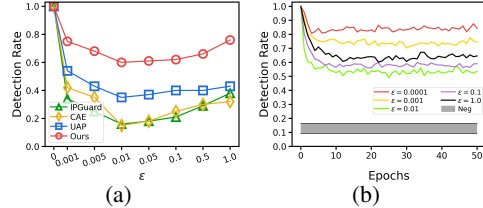

(a)    (b)

Figure 7: The results for adversarial training on CIFAR-100, varying (a) perturbation budget and (b) training epochs.

Table 3: Performance under three model extraction attacks.

| Attack | Dataset | Test Accuracy | F1-score | | | |
|---|---|---|---|---|---|---|
| | | | IPGuard | CAE | UAP | Ours |
| PRADA | CIFAR-100 | 0.721 | 0.70 | 0.76 | 0.85 | **0.90** |
| | ImageNet | 0.743 | 0.59 | 0.63 | 0.72 | **0.83** |
| Knockoff | CIFAR-100 | 0.734 | 0.69 | 0.84 | 0.87 | **0.93** |
| | ImageNet | 0.764 | 0.62 | 0.68 | 0.75 | **0.86** |
| HL | CIFAR-100 | 0.723 | 0.35 | 0.38 | 0.51 | **0.67** |
| | ImageNet | 0.735 | 0.29 | 0.27 | 0.44 | **0.58** |

model for 20 epochs ($\epsilon = 1.0$ is quite large for adversarial training but we still test with this value). Figure 7a illustrates the results for CIFAR-100. Interestingly, we observe a trend where the detection rate first decreases (from 1.0 to 0.60) and then increases (from 0.60 to 0.76), as $\epsilon$ rises from 0.001 to 1.0. A perturbation budget of 0.01 achieves the best attack result, rendering $40\%$ of fingerprinting samples invalid. This implies that an attacker does not necessarily have to impose adopt higher $\epsilon$ to launch stronger attacks. For $\epsilon = 0.01$, our approach successfully detects $60\%$ of trajectories, which are about 3 times that of IPGuard and CAE, and 1.5 times that of UAP, demonstrating strong robustness against adversarial training. Additionally, we record the results during 50 epochs of adversarial training in Figure 7b. The first 5 epochs witness a big drop of fingerprint detection rate. As the number of training epochs continues to increase, the fingerprint detection rates of these models remains almost unchanged. We speculate that the models have already attained the $\epsilon-$adversarial robustness during the first few training epochs. As such, subsequent training does not further decrease the effectiveness of fingerprinting samples.

**Impact of Model Extraction Attacks.** Since the fingerprint detection rate alone cannot reflect the gap between positive and negative models, here, we calculate the F1-score of correctly identified fingerprint samples under PRADA and Knockoff attacks. From the results in Table 3, we can observe that our method achieves the best results, leading the second-place method (UAP) by nearly 0.1 under all attacks. It is worth noting that better extraction performance (higher test accuracy) is inevitably accompanied by a substitute model that learns a decision surface more similar to the source model. This also leads to a better transfer of the fingerprint space (a part of the decision surface) as well. Moreover, as a data-free model stealing attack with access to only hard labels, HL distills a relatively more dissimilar substitute model compared to the source model under less prior knowledge, resulting in the lowest F1-score among the four fingerprinting methods.

# 6 Conclusion

In this paper, we have presented ADV-TRA, a robust fingerprinting scheme that leverages adversarial trajectories to fingerprint the DNN model. ADV-TRA generates a chain of samples with varying levels of adversarial perturbation, and further extends to the surface trajectory that involves a series of fixed-length trajectories with dynamically adjusted step sizes. By doing so, the alteration in the decision boundary can be captured more accurately, and the misjudgment caused by local resemblance from innocent models can be significantly reduced. Extensive evaluation results on three datasets demonstrate that ADV-TRA is able to defend against various removal attacks even under severe attack intensities, exhibiting greatly superior robustness and lower false positive rates compared to existing state-of-the-art fingerprinting methods.

## Acknowledgement

This work was supported in part by the National Natural Science Foundation of China under Grants 62272183 and 62171189; by the National Key Technology R&D Program of China under Grant 2022YFA0911800; by the Key R&D Program of Hubei Province under Grants 2024BAA008, 2024BAA011, 2024BAB016, 2024BAB031, 2023BAB074; by the special fund for Wuhan Artificial Intelligence Innovation under Grant 2022010702040061; and by the Open Project Funding of the Key Laboratory of Intelligent Sensing System and Security, Ministry of Education under Grant KLISSS202401.

## Footnotes

[2]Here, the term "progressively adversarial samples" refers to a series of samples with progressively increasing adversarial perturbations, rather than the adversarial sample in the strict sense.

[3]The auxiliary data can be collected from the wild or internet with the same distribution as the task do-

[5]http://www.cs.toronto.edu/~kriz/cifar.html

[6]`https://image-net.org/`

[7]`https://pytorch.org/vision/stable/models.html#classification`

[8]`https://github.com/BorealisAI/advertorch`

[9]https://github.com/somepago/dbViz

[10]https://www.anaconda.com/download/

## References

[1] Guibiao Liao, Jiankun Li, and Xiaoqing Ye. Vlm2scene: Self-supervised image-text-lidar learning with foundation models for autonomous driving scene understanding. In *Proceedings of AAAI*, pages 3351–3359, 2024.

[2] Jiun Tian Hoe, Xudong Jiang, Chee Seng Chan, Yap-Peng Tan, and Weipeng Hu. Interactdiffusion: Interaction control in text-to-image diffusion models. In *Proceedings of IEEE/CVF CVPR*, 2024.

[3] Jun Shi, Shulan Ruan, Ziqi Zhu, Minfan Zhao, Hong An, Xudong Xue, and Bing Yan. Predictive accuracy-based active learning for medical image segmentation. In *Proceedings of IJCAI*, 2024.

[4] Jonathan Vanian and Kif Leswing. Chatgpt and generative ai are booming, but the costs can be extraordinary. https://www.cnbc.com/2023/03/13/chatgpt-and-generative-ai-are-booming-but-at-a-very-expensive-price.html. 2023-03-13.

[5] Yuzheng Wang, Zhaoyu Chen, Dingkang Yang, Pinxue Guo, Kaixun Jiang, Wenqiang Zhang, and Lizhe Qi. Out of thin air: Exploring data-free adversarial robustness distillation. In *Proceedings of AAAI*, pages 5776–5784, 2024.

[6] Yossi Adi, Carsten Baum, Moustapha Cisse, Benny Pinkas, and Joseph Keshet. Turning your weakness into a strength: Watermarking deep neural networks by backdooring. In *Proceedings of USENIX Security Symposium*, pages 1615–1631, 2018.

[7] Lixin Fan, Kam Woh Ng, and Chee Seng Chan. Rethinking deep neural network ownership verification: Embedding passports to defeat ambiguity attacks. In *Proceedings of NeurIPS*, 2019.

[8] Hengrui Jia, Christopher A Choquette-Choo, Varun Chandrasekaran, and Nicolas Papernot. Entangled watermarks as a defense against model extraction. In *Proceedings of USENIX Security Symposium*, pages 1937–1954, 2021.

[9] Nils Lukas, Yuxuan Zhang, and Florian Kerschbaum. Deep neural network fingerprinting by conferrable adversarial examples. In *Proceedings of ICLR*, 2021.

[10] Zirui Peng, Shaofeng Li, Guoxing Chen, Cheng Zhang, Haojin Zhu, and Minhui Xue. Fingerprinting deep neural networks globally via universal adversarial perturbations. In *Proceedings of IEEE/CVF CVPR*, pages 13430–13439, 2022.

[11] Jialuo Chen, Jingyi Wang, Tinglan Peng, Youcheng Sun, Peng Cheng, Shouling Ji, Xingjun Ma, Bo Li, and Dawn Song. Copy, right? A testing framework for copyright protection of deep learning models. In *Proceedings of IEEE S&P*, pages 824–841, 2022.

[12] Jiyang Guan, Jian Liang, and Ran He. Are you stealing my model? Sample correlation for fingerprinting deep neural networks. In *Proceedings of NeurIPS*, pages 36571–36584, 2022.

[13] Xudong Pan, Mi Zhang, Yifan Yan, Yining Wang, and Min Yang. Cracking white-box dnn watermarks via invariant neuron transforms. In *Proceedings of ACM SIGKDD*, pages 1783–1794, 2023.

[14] Xiaoyu Cao, Jinyuan Jia, and Neil Zhenqiang Gong. IPGuard: Protecting intellectual property of deep neural networks via fingerprinting the classification boundary. In *Proceedings of ACM ASIACCS*, pages 14–25, 2021.

[15] Han Qiu, Yi Zeng, Shangwei Guo, Tianwei Zhang, Meikang Qiu, and Bhavani Thuraisingham. Deepsweep: An evaluation framework for mitigating dnn backdoor attacks using data augmentation. In *Proceedings of ACM ASIACCS*, pages 363–377, 2021.

[16] Wei Zong, Yang-Wai Chow, Willy Susilo, Joonsang Baek, Jongkil Kim, and Seyit Camtepe. IPRemover: A generative model inversion attack against deep neural network fingerprinting and watermarking. In *Proceedings of AAAI*, pages 7837–7845, 2024.

[17] Gowthami Somepalli, Liam Fowl, Arpit Bansal, Ping Yeh-Chiang, Yehuda Dar, Richard Baraniuk, Micah Goldblum, and Tom Goldstein. Can neural nets learn the same model twice? Investigating reproducibility and double descent from the decision boundary perspective. In *Proceedings of IEEE/CVF CVPR*, pages 13699–13708, 2022.

[18] Hengrui Jia, Hongyu Chen, Jonas Guan, Ali Shahin Shamsabadi, and Nicolas Papernot. A zest of lime: Towards architecture-independent model distances. In *Proceedings of ICLR*, 2022.

[19] Ian J Goodfellow, Jonathon Shlens, and Christian Szegedy. Explaining and harnessing adversarial examples. In *Proceedings of ICLR*, 2015.

[20] Jie Wan, Jianhao Fu, Lijin Wang, and Ziqi Yang. BounceAttack: A query-efficient decision-based adversarial attack by bouncing into the wild. In *Proceedings of IEEE S&P*, 2024.

[21] Honglin Li, Chenglu Zhu, Yunlong Zhang, Yuxuan Sun, Zhongyi Shui, Wenwei Kuang, Sunyi Zheng, and Lin Yang. Task-specific fine-tuning via variational information bottleneck for weakly-supervised pathology whole slide image classification. In *Proceedings of IEEE/CVF CVPR*, pages 7454–7463, 2023.

[22] Ning Liu, Xiaolong Ma, Zhiyuan Xu, Yanzhi Wang, Jian Tang, and Jieping Ye. Autocompress: An automatic dnn structured pruning framework for ultra-high compression rates. In *Proceedings of AAAI*, pages 4876–4883, 2020.

[23] Alireza Ganjdanesh, Shangqian Gao, and Heng Huang. Jointly training and pruning cnns via learnable agent guidance and alignment. In *Proceedings of IEEE/CVF CVPR*, 2024.

[24] Ali Shafahi, Mahyar Najibi, Mohammad Amin Ghiasi, Zheng Xu, John Dickerson, Christoph Studer, Larry S Davis, Gavin Taylor, and Tom Goldstein. Adversarial training for free! In *Proceedings of NeurIPS*, 2019.

[25] Xiaoling Zhou, Wei Ye, Zhemg Lee, Rui Xie, and Shikun Zhang. Boosting model resilience via implicit adversarial data augmentation. In *Proceedings of IJCAI*, 2024.

[26] Mika Juuti, Sebastian Szyller, Samuel Marchal, and N Asokan. PRADA: Protecting against DNN model stealing attacks. In *Proceedings of IEEE S&P*, pages 512–527, 2019.

[27] Tribhuvanesh Orekondy, Bernt Schiele, and Mario Fritz. Knockoff nets: Stealing functionality of black-box models. In *Proceedings of IEEE/CVF CVPR*, pages 4954–4963, 2019.

[28] Nicolas Papernot, Patrick McDaniel, Ian Goodfellow, Somesh Jha, Z Berkay Celik, and Ananthram Swami. Practical black-box attacks against machine learning. In *Proceedings of ACM ASIACCS*, pages 506–519, 2017.

[29] Alexey Kurakin, Ian J Goodfellow, and Samy Bengio. Adversarial machine learning at scale. In *Proceedings of ICLR*, 2016.

[30] Alex Krizhevsky, Vinod Nair, and Geoffrey Hinton. Cifar-10 (canadian institute for advanced research), 2010.

[31] Jia Deng, Wei Dong, Richard Socher, Li-Jia Li, Kai Li, and Li Fei-Fei. ImageNet: A large-scale hierarchical image database. In *Proceedings of IEEE/CVF CVPR*, pages 248–255, 2009.

[32] Kang Yang, Run Wang, and Lina Wang. Metafinger: Fingerprinting the deep neural networks with meta-training. In *Proceedings of IJCAI*, 2022.

[33] Song Han, Jeff Pool, John Tran, and William Dally. Learning both weights and connections for efficient neural network. In *Proceedings of NeurIPS*, 2015.

[34] Aleksander Madry, Aleksandar Makelov, Ludwig Schmidt, Dimitris Tsipras, and Adrian Vladu. Towards deep learning models resistant to adversarial attacks. In *Proceedings of ICLR*, 2018.

[35] Sunandini Sanyal, Sravanti Addepalli, and R Venkatesh Babu. Towards data-free model stealing in a hard label setting. In *Proceedings of IEEE/CVF CVPR*, pages 15284–15293, 2022.

[36] Nils Lukas, Edward Jiang, Xinda Li, and Florian Kerschbaum. Sok: How robust is image classification deep neural network watermarking? In *Proceedings of IEEE S&P*, pages 787–804, 2022.

[37] Zehao Tian, Zixiong Wang, Ahmed M Abdelmoniem, Gaoyang Liu, and Chen Wang. Knowledge representation of training data with adversarial examples supporting decision boundary. *IEEE Transactions on Information Forensics and Security*, 18:4116 – 4127, 2023.

[38] Ziheng Huang, Boheng Li, Yan Cai, Run Wang, Shangwei Guo, Liming Fang, Jing Chen, and Lina Wang. What can discriminator do? Towards box-free ownership verification of generative adversarial network. In *Proceedings of IEEE/CVF CVPR*, pages 5009–5019, 2023.

[39] Thibault Maho, Teddy Furon, and Erwan Le Merrer. FBI: Fingerprinting models with benign inputs. *IEEE Transactions on Information Forensics and Security*, 18:5459–5472, 2023.

[40] Xudong Pan, Yifan Yan, Mi Zhang, and Min Yang. MetaV: A meta-verifier approach to task-agnostic model fingerprinting. In *Proceedings of ACM SIGKDD*, pages 1327–1336, 2022.

[41] Yue Zheng, Si Wang, and Chip-Hong Chang. A dnn fingerprint for non-repudiable model ownership identification and piracy detection. *IEEE Transactions on Information Forensics and Security*, 17:2977–2989, 2022.

[42] Tian Dong, Shaofeng Li, Guoxing Chen, Minhui Xue, Haojin Zhu, and Zhen Liu. RAI2: Responsible identity audit governing the artificial intelligence. In *Proceedings of NDSS*, 2023.

[43] Yuchen Sun, Tianpeng Liu, Panhe Hu, Qing Liao, Shouling Ji, Nenghai Yu, Deke Guo, and Li Liu. Deep intellectual property: A survey. *CoRR, arXiv: 2304.14613*, 2023.

[44] Jangho Kim, Jayeon Yoo, Yeji Song, KiYoon Yoo, and Nojun Kwak. Finding efficient pruned network via refined gradients for pruned weights. In *Proceedings of ACM MM*, pages 9003–9011, 2023.

[45] Yizhen Yuan, Rui Kong, Shenghao Xie, Yuanchun Li, and Yunxin Liu. Patchbackdoor: Backdoor attack against deep neural networks without model modification. In *Proceedings of ACM MM*, pages 9134–9142, 2023.

[46] Meiqi Wang, Han Qiu, Tianwei Zhang, Meikang Qiu, and Bhavani Thuraisingham. Mitigating query-based neural network fingerprinting via data augmentation. *ACM Transactions on Sensor Networks*, 2023.

[47] Zhengyuan Jiang, Jinghuai Zhang, and Neil Zhenqiang Gong. Evading watermark based detection of ai-generated content. In *Proceedings of ACM CCS*, pages 1168–1181, 2023.

[48] Laurens Van der Maaten and Geoffrey Hinton. Visualizing data using t-SNE. *Journal of Machine Learning Research*, 9(86):2579–2605, 2008.

[49] Roland S Zimmermann, Wieland Brendel, Florian Tramer, and Nicholas Carlini. Increasing confidence in adversarial robustness evaluations. In *Proceedings of NeurIPS*, pages 13174–13189, 2022.

[50] Mingxing Duan, Yunchuan Qin, Jiayan Deng, Kenli Li, and Bin Xiao. Dual attention adversarial attacks with limited perturbations. *IEEE Transactions on Neural Networks and Learning Systems*. doi: 10.1109/TNNLS.2023.3274142.

[51] Junyoung Byun, Seungju Cho, Myung-Joon Kwon, Hee-Seon Kim, and Changick Kim. Improving the transferability of targeted adversarial examples through object-based diverse input. In *Proceedings of IEEE/CVF CVPR*, pages 15244–15253, 2022.

[52] Shangbo Wu, Yu-an Tan, Yajie Wang, Ruinan Ma, Wencong Ma, and Yuanzhang Li. Towards transferable adversarial attacks with centralized perturbation. In *Proceedings of AAAI*, pages 6109–6116, 2024.

[53] Adam Paszke, Sam Gross, Francisco Massa, Adam Lerer, James Bradbury, Gregory Chanan, Trevor Killeen, Zeming Lin, Natalia Gimelshein, Luca Antiga, et al. Pytorch: An imperative style, high-performance deep learning library. In *Proceedings of NeurIPS*, 2019.

# Appendix

In this appendix, we present the following additional contents: (1) A review of model fingerprinting methods and fingerprint removal attacks (Appendix A); (2) The pseudocode of our ADV-TRA (Appendix B); (3) Detailed settings for our experimental setups (Appendix C); (4) Additional experimental results, including robustness analysis with decision boundary visualization (Appendix D.1), comparison of the surface trajectory and the bilateral trajectory (Appendix D.2), ablation studies analyzing the impact of various hyperparameters. (Appendix D.3), the visualization of adversarial trajectories (Appendix D.4), and time overheads (Appendix D.5); (5) Discussion of limitations and the future work (Appendix E); (6) Technical details about our experiment environment (Appendix F.1) and the instruction for the code of our method (Appendix F.2).

## A  Related Work

### A.1  Model Fingerprinting

In order to protect the IP of DNN models, model fingerprinting extracts the features of a model in a non-invasive way, usually by detecting the decision boundaries [37].

Most fingerprinting approaches utilize adversarial samples to probe the decision boundaries of the models. Cao et al. [14] argue that a model can be uniquely represented by its decision boundary. Hence, they find some special data points (i.e., adversarial samples) near the boundary to mark the target model, which is the first model fingerprinting method. Later, several studies are working to explore the transferability of adversarial samples. For instance, Lukas et al. [9] design special adversarial samples that show strong transferability for homology models. They also study the impact of some factors (e.g., model architecture and removal attacks) that affect such transferability. Lukas et al. [10] also point out that prior fingerprinting methods still suffer a high false positive rate: two unrelated models always have the same fingerprinting samples due to the adversarial transferring. Instead, they generate Universal Adversarial Perturbations to match the model. Any normal sample added with such perturbation would shift to the decision boundary. However, from the perspective of ownership verification, this kind of perturbation lacks security and uniqueness [6, 38]: Once the attacker knows the pattern of perturbation, he can easily invalidate all fingerprinting samples by removing the perturbation from them.

In addition to adversarial-based model fingerprinting, some studies use the intermediate computations to measure the model similarity. For example, Maho et al. [39] design a greedy algorithm that leverages few benign inputs and C.E. Shannon's information theory to quantify the statistical similarity between the internal outputs of two models. Guan et al. [12] select samples with inconsistent prediction results across two groups of reference models and utilize their pairwise relationships for suspicious model identity recognition. Besides, the model internal parameters can also serve as fingerprints. Similarly, Jia et al. [18] construct a reference dataset to train a linear model, and compute the cosine distance between the weights of the linear models. Zheng et al. [41] project the front-layer weights onto a random space defined by the model owner's identity, allowing non-repudiable and irrevocable ownership proof against model IP misappropriation and ownership fraud. Recent works [11, 42] build an audit framework to assess the infringement of model IP. The former leverages multiple testing metrics (e.g., neuron output distance) to calculate the similarity of two models. The latter projects the high-dimensional model parameters into low-dimensional data and compares their distributions.

For model fingerprinting, one open question is how to extract fingerprints that satisfies both uniqueness (low false positive rate) and robustness (robust against removal attacks) [43]. Current fingerprinting schemes, which verify with a special type of adversarial samples individually, are less than satisfactory, especially for removal attacks like adversarial training [25, 36]. In this paper, we attempt to generate a chain of samples that have intrinsic connections with each other, using them to capture more comprehensive characteristics of the decision surface to enhance both the uniqueness and robustness.

### A.2  Removal Attacks

Several studies are also working on exploring the vulnerabilities of model fingerprinting through removal attacks. Removal attacks involve modifying the model, including parameter modifications

such as fine-tuning [21] or structural modifications such as pruning [44]. Such modifications can remove or alter the original fingerprints of the model, leading to the failure of model identification.

Even though many works [14, 9, 10, 11] on model fingerprinting report their robustness against a variety of removal attacks, the strength of the simulated removal attacks in these works is relatively weak. A recent study [36] makes a comprehensive robustness evaluation for existing watermarking and fingerprinting methods, and points out that none of the surveyed schemes is robust in practice. Yang et al. [32] emphasize that changes in a model's decision boundaries caused by model modification could render fingerprinting methods less reliable, particularly for input modification [45] and adversarial training [25]. Another kind of removal attack originates from model extraction [5], where an attacker can distill knowledge from the source model and train a substitute model. PRADA [26] aims to steal black-box deployed victim models, with the objective of obtaining a substitute model with strong transferability of adversarial samples. Apart from the aforementioned model modification attacks, Wang et al. [46] also find randomized transformations to input samples can significantly undermine query-based fingerprinting methods. Likewise, Jiang et al. [47] propose an adversarial post-processing method to evade copyright detection of the model's products.

In this paper, we select four representative removal attacks, including three approaches capable of altering model decision boundaries (fine-tuning, pruning, and adversarial training), as well as three model extraction attacks, to evaluate the robustness of model fingerprinting methods. We also strengthen the intensity of the removal attacks to simulate a more powerful attacker that may occur in real-world scenarios.

## B Pseudocode for ADV-TRA

---
**Algorithm 1** ADV-TRA

---
**Input**: Source model $f_{src}$, base sample $x_0$, number of classes $m$, length of trajectory $l$, number of generation iteration $t$.
**Output**: Surface trajectory $\mathcal{T}_{surf}$.

    Randomly choose pre-defined classes $\mathbf{c} = \{c_1, ..., c_m\}$
    Let $\mathcal{T}_{surf} = \varnothing$
    **for** $c_j$ in $\mathbf{c}$ **do**
        **for** $epoch = 1$ to $t$ **do**
            Initialize step size of each step $\mathbf{s} = \{s_0, s_1, \ldots, s_{l-1}\}$         ▷ Trajectory initialization
            $\mathbf{s} \leftarrow \text{LengthControl}(f_{src}, \mathbf{s}, x_0, c_j, \alpha_{lc})$
            Calculate $\mathcal{L}_{brk}(\mathbf{s})$ and $\mathcal{L}_{fwd}(\mathbf{s})$
            Optimize $\mathbf{s}$ with $\mathcal{L}_{fwd}$ and $\mathcal{L}_{brk}$         ▷ Boundary Probing
        **end for**
        $\mathcal{T}_{bi} = \mathcal{T}(f_{src}, x_0, c_j, \mathbf{s} \cup \hat{\mathbf{s}})$         ▷ Trajectory Bilateralization
        $\mathcal{T}_{surf} \leftarrow \mathcal{T}_{surf} \cup \mathcal{T}_{bi}$         ▷ Trajectory Connection
        Update $x_0$ with the last sample of $\mathcal{T}_{bi}$
    **end for**
    **return** $\mathcal{T}_{surf}$

---

## C Experiment Setup

**Datasets.** We consider three commonly used image datasets CIFAR-10 [30], CIFAR-100 [30], and ImageNet [31] in our experiments.

- **CIFAR-10**[5]. This dataset is a widely used benchmark dataset in the field of computer vision and machine learning. It consists of $60,000$ $32 \times 32$ images in 10 classes, with $6,000$ images per class.

- **CIFAR-100**[5]. This dataset consists of $60,000$ color images of $100$ classes containing $600$ images each. It has the same image size as CIFAR-10 but contains 20 superclasses instead of 10 classes, with each super class having 5 sub-classes.

- **ImageNet**[6]. This dataset is a large-scale dataset containing 14 million $224 \times 224$ images of $1,000$ classes. It continues to be one of the most popular and challenging datasets for benchmark classification and detection models.

  For both CIFAR-10 and CIFAR-100, we allocate $50,000$ samples for training the model, while reserving the remaining $10,000$ samples for the attacker to launch removal attacks. For ImageNet, considering the huge computation cost for training a model and suspect models from scratch, we utilize pre-trained models available in Pytorch[7]. We allocate $60,000$ samples to launch removal attacks.

**Models.** For the source models, we consider ResNet20 for CIFAR-10, WideResNet for CIFAR-100, and ResNet50 for ImageNet. Specifically, for the negative models from DAST and DADT, we widely adopted other model architectures, including 6 different structures: VGG16, ResNet152, DenseNet201, EfficientNetV2, InceptionV3, and MobileNetV3.

**Existing Defenses.** We compare ADV-TRA with the following four fingerprinting methods against the removal attacks: (1) **CoRt** [11] generates a series of adversarial examples using FGSM [19] and PGD [34]. (2) **IPGuard** [14] designs a type of fingerprinting samples that is superior to general adversarial samples and can better locate the decision boundaries. (3) **CAE** [9] proposes to use multiple surrogate models to find the conferrable fingerprinting samples that demonstrate better transferability on stolen models. (4) Given any normal sample, **UAP** [10] designs a special adversarial perturbation that only reacts to the source model.

**Implementation Details.** In general, we train the source model and negative models for 200 epochs with a batch size $b = 128$. We use SGD optimizer with learning rate of $0.1$, momentum of $0.9$, and weight decay of 5e-4. Additionally, we adopt a learning rate scheduler to decay the learning rate by $0.1\times$ at epochs 60, 120, and 160. For all fingerprinting methods, we select 100 clean samples to generate fingerprinting samples. For our ADV-TRA, we also choose 100 (clean) base samples to generate 100 trajectories (the number of $\mathcal{T}_{bi}$ $m = 10$ (in Appendix D.2, we discuss the impact of the $m$ on performance) for CIFAR-100 and ImageNet; the length of $\mathcal{T}_{int}$ $l = 2$), and set the threshold for mutation rate $Thr_{mut} = 0.5$, the brake factor $\alpha_{brk} = 0.9$. For each set of experiments, we build a distinct source model for every dataset. Within each set, we perform five trials and then compute the average results.

**Removal Attacks.** In this paper, we adopt four common removal attacks to evaluate the robustness of fingerprinting methods.

- **Fine-Tuning.** Fine-tuning uses auxiliary data to continue to train a pre-trained model (source model). We adopt four common strategies as in [6, 32]: (1) Fine-Tune Last Layer (**FTLL**): only update the last layer while training; (2) Fine-Tune All Layers (**FTAL**): update all layers while training; (3) Retrain Last Layer (**RTLL**): re-initialize the last layer before FTLL; and (4) Retrain All Layers (**RTAL**): re-initialize the last layer before FTAL. We strengthen the attack intensity by lengthening the fine-tuning epoch as well as increasing the learning rate. Specifically, we apply SGD optimizer with a learning rate $\eta = 0.001$ for FTLL and FTAL attack. For RTLL and RTAL attacks, we utilize SGD optimizer with a learning rate of $0.01$, momentum of $0.9$, and weight decay of 5e-4. We set the epochs of all fine-tuning strategies to 50.

- **Pruning.** Pruning is used for model compression, so that the model's memory for fingerprints diminishes over the pruning process. We utilize the technique in [33] for our evaluation. We vary the pruning rate $p$ from $0.1$ to $0.9$, i.e., pruning $p$ fraction of parameters which have the smallest absolute values. After pruning, we fine-tune the model for 10 epochs for CIFAR-10 and CIFAR-100, and 20 epochs for ImageNet, in order to ensure the accuracy of the model.

- **Adversarial Training.** Adversarial training helps models withstand misleading inputs by incorporating them during the training process to improve the model's robustness. We use the method in [34] in our experiments. We adopt standard $\ell_\infty$ PGD training, which has been proved to achieve the best empirical robustness. In detail, we leverage advertorch[8] library to generate specified adversarial samples. We vary the perturbation budget $\epsilon$ from 1e-4 to 1.0 to generate adversarial samples used for adversarial training. To avoid compromising the model's accuracy,

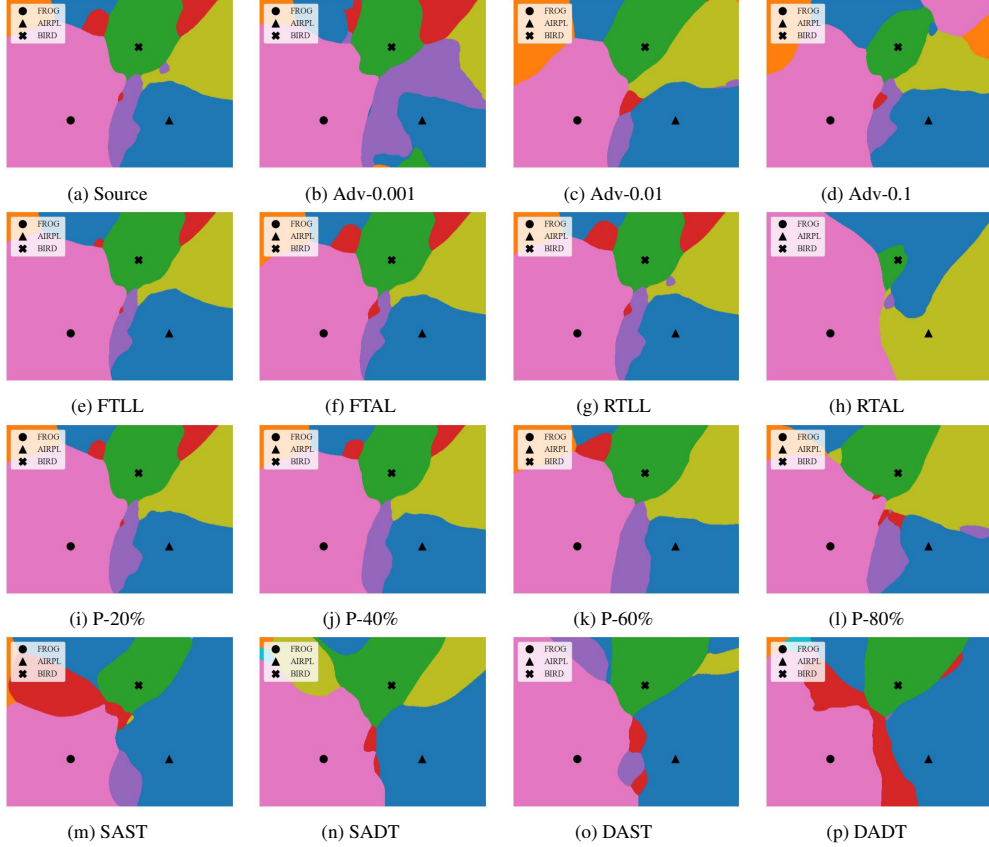

|                |                |                |                |
|----------------|----------------|----------------|----------------|
| (a) Source     | (b) Adv-0.001  | (c) Adv-0.01   | (d) Adv-0.1    |
| (e) FTLL       | (f) FTAL       | (g) RTLL       | (h) RTAL       |
| (i) P-20%      | (j) P-40%      | (k) P-60%      | (l) P-80%      |
| (m) SAST       | (n) SADT       | (o) DAST       | (p) DADT       |

Figure 8: Visualization for the decision surface under various removal attacks. Each color represents a class.

we incorporate normal samples during adversarial training. Additionally, we enhance the attack intensity by prolonging the training epochs to 50.

- **Model Extraction Attack.** Model extraction enables an attacker to derive a substitute model from the source model. We apply three commonly used attacks, namely, PRADA [26], Knock-off [27] and HL [35] for evaluation. PRADA is a benchmark extraction attack in model IP protection, which selects specific hyperparameters and generates synthetic data to train a substitute model with high watermarking transferability. Knockoff collect data from different domains and train a substitute model with fewer queries in a reinforcement learning manner. As a data-free model stealing attack, HL leverages the structure of DCGAN and alternately trains a substitute model and generator in a hard label setting. All extraction attacks adopt the same model architecture as the source model.

**Metrics.** In our experiments, we use *Fingerprint Detection Rate* as the main evaluation metric, which is widely used in previous studies [14, 10, 36]. This metric represents the proportion of fingerprinting samples (adversarial trajectories in our case) matched with a suspect model. We also employ the Receiver Operating Characteristic (ROC) curve, Area Under the Curve (AUC), T1%F, and T10%F as the evaluation metrics. The ROC curve plots the True Positive Rate (TPR) against the False Positive Rates (FPR) at various threshold settings. T1%F and T10%F represent the TPR values at 1% and 10% FPR, respectively, with higher values indicating better verification performance. These metrics can be used to measure the ability of the fingerprinting samples to accurately distinguish the positive models from the negative ones. To measure the impact of removal attacks on decision boundaries, we compute the average mutation rate $r_{mut}$ across multiple suspect models in Table 1, represented as $\bar{r}_{mut}$, which quantifies the discrepancy in decision boundaries between the suspect and source models.

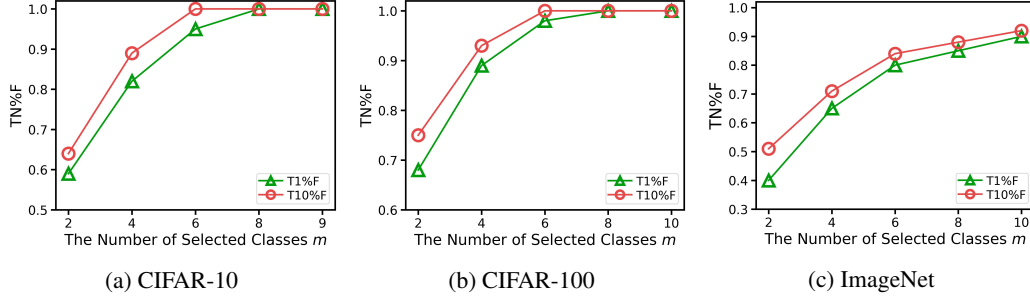

|                | (a) CIFAR-10 | (b) CIFAR-100 | (c) ImageNet |
|----------------|--------------|---------------|--------------|

Figure 9: T1%F and T10%F with the varying number of classes spanned by trajectories. Note that when the number of classes spanned by a trajectory is two, it is equivalent to a bilateral trajectory as defined in Section 4.1 in the main text.

# D   Additional Experimental Results

## D.1   Robustness Analysis with Decision Boundary Visualization

To better understand how the removal attacks affect the source model from the perspective of the decision boundary, we leverage a recent model visualization technique [17] based on data manifold[9]. This method requires three base images to construct the decision boundary surface. We randomly select three samples belonging to the classes FROG, AIRPL, and BIRD from CIFAR-10 dataset, and the results are presented in Figure 8. It can be found that stronger attacks (Adv-0.01, RTAL, and P-80%) induce greater changes to the decision surface, which is consistent with their relatively lower fingerprint detection rates shown in Table 1 in the main text. Compared to the source model, the Adv-0.01 model exhibits a reduction in the number of regions corresponding to the CAT class (red region) from 3 to 1. Additionally, there is a significant increase in the area corresponding to the DEER class (orange region). It can also be observed that RTAL greatly alters the decision surface by eliminating most of the smaller regions. In general, the source model is more vulnerable to adversarial training attacks. For negative models, even though the SAST model adopts the same training data and model architecture as the source model, the differences between its decision surface and the source model's are still greater compared to the positive models. This is why model fingerprinting is able to distinguish between infringing positive models and innocent negative models. It can also be observed that there is an obvious inductive bias of the model architecture from DAST and DADT models.

## D.2   Surface Trajectory vs. Bilateral Trajectory

In this part, we investigate the advantages of the surface trajectory over the bilateral trajectory. We evaluate the ability of multi-class trajectories to capture global fingerprint features on 100 suspect models (same setup as Section 5.1.), assessing whether traversing more classes can help reduce the misjudgment of trajectories. To ensure a fair comparison across different experimental groups and avoid the impact of threshold selection, we utilize the T1%F and T10%F metrics. The results are shown in Figure 9.

Obviously, a trajectory crossing only two classes (corresponding to the bilateral trajectory) is insufficient to adequately extract the "unique" or "global" fingerprint features of the model, and its performance may even be inferior to that of single-point fingerprinting methods (e.g., UAP, see Table 2 in the main text). However, when the number of crossed classes increases from 2 to 6, both T1%F and T10%F show significant improvements, strongly implying a sharp fall in the false positive rate. For CIFAR-10 and CIFAR-100 datasets, T1%F and T10%F approach 1.0 once the number of crossed classes exceeds 6, whereas for ImageNet dataset, T1%F and T10%F continue to increase slowly as the number of spanning classes increases. This may be attributed to the large number of classes in ImageNet dataset, which has a more complex decision surface and a higher level of intricacy in the fingerprint patterns. As mentioned in Section 4.1 in the main text, a bilateral trajectory $\mathcal{T}_{bi}$ only attends the decision boundary between two classes, which is still restricted to local fingerprinting. In contrast, a surface trajectory $\mathcal{T}_{surf}$ that spans multiple classes has the potential to

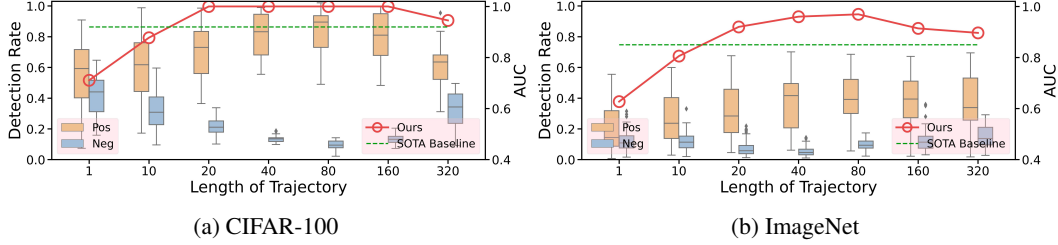

Figure 10: The distribution and AUC value of fingerprint detection rate for two types of suspect models under different trajectory lengths. Length=1 corresponds to the single-point fingerprinting method. We choose UAP as the baseline in view of its excellent performance.

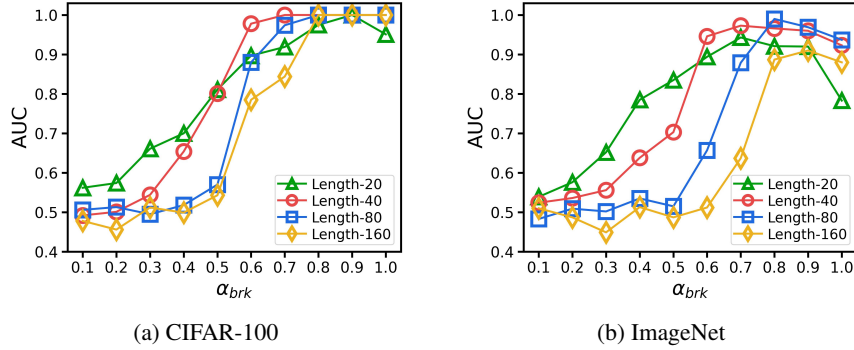

Figure 11: Performance over different brake factor $\alpha_{brk}$ for various trajectory lengths.

capture richer and higher-level features of decision surface, thereby significantly reducing the false positives of innocent models.

## D.3 Impact of Parameters

### D.3.1 Impact of Trajectory Length

The length of adversarial trajectories is a critical hyperparameter that determines the effectiveness and stability of ADV-TRA. In this part, we vary the length of trajectories from 10 to 320. Here, a length of 320 signifies that for a trajectory traversing 10 classes, there would be 32 samples per class.

As shown in Figure 10, as the length increases from 10 to 80, the gap between positive models and negative models gradually widens until completely separated, and the range of negative models shrinks from 0.597 to 0.095. However, when the length exceeds 80, the trend is reversed. When the length is 320, we can see an obvious breakdown. The AUC value maintains 1.0 within the length ranging from 20 to 180. Even when the AUC drops to 0.942 at the length of 320, it is 0.03, still higher than the baseline (0.91). Overall, for lengths surpassing 20, the effectiveness of our method is distinctly superior to the baseline.

Since the distance from a sample to the decision boundary is fixed, a longer length means smaller step sizes and more samples in the trajectory. Ideally, a trajectory with more samples could provide richer details regarding the decision boundary. However, there is a convergence problem during trajectory generation: when the step size is too small, it may get trapped in local optima, rendering the trajectory difficult to cross over the decision boundary. We observe that some of the trajectories do exhibit this phenomenon when the length is 320. This is a common problem for gradient-based optimization algorithms. Therefore it is important to determine the length of the trajectory. Based on the experimental results, setting the length between 20 to 80 is a wise choice.

### D.3.2 Impact of Brake Factor

Another key point is the brake factor $\alpha_{brk}$, which regulates the proportional relationship between the step sizes of two adjacent steps in the trajectory. A small $\alpha_{brk}$ may result in subsequent excessively

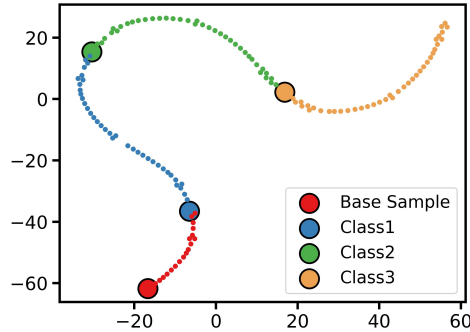

Figure 12: $t$-SNE visualization of the adversarial trajectory. It depicts an adversarial trajectory traversing four classes starting from a base sample (red), where larger points denote the first samples crossing the decision boundary of a new class. We sample a base sample from class $0$ and guide it through class $1$, $2$, and $3$.

small step size, getting trapped in local optima. On the contrary, a large $\alpha_{brk}$ may fail to capture the rich information around the decision boundary.

As shown in Figure 11, we can clearly see that both $\alpha_{brk}$ and the length jointly affect the AUC value. When the $\alpha_{brk}$ is small, the effectiveness of the verification dramatically decreases and becomes almost equivalent to random guessing. Interestingly, shorter trajectories tend to perform better under extremely low $\alpha_{brk}$ values. In other words, shorter trajectories show greater tolerance for smaller $\alpha_{brk}$, as step sizes decrease exponentially along the trajectory. For example, the AUC curve for a length of $40$ begins to experience substantial growth around $\alpha_{brk} = 0.3$, whereas the curve for a length of $160$ does not increase until $\alpha_{brk} = 0.6$. It can also be obviously seen that the peak of the AUC curve occurs at $\alpha_{brk} < 1$ ($\alpha_{brk} = 1$ corresponds to a fixed step size). This is more pronounced on ImageNet dataset, which demonstrates that the strategy of decaying step size does help to attend more local details of decision boundaries.

### D.4 Visualization

Through $t$-SNE [48] visualization technique, we visualize our adversarial trajectory fingerprinting in Figure 12. It can be seen that an adversarial trajectory contains multiple sub-fingerprints (i.e., bilateral trajectories) with varying levels of perturbations. The trajectory starts from a base sample, and progresses toward specified classes until crossing multiple classes. This process generates a series of coherent sub-fingerprints. The number of sub-fingerprints contained in each class is fixed, thus realizing the trajectory of a fixed length, which makes the required query number limited. Furthermore, the progressiveness of the trajectory enables it to mark the positioning of decision boundaries (i.e., where the larger point is located), thereby enabling measurement of how much the decision boundaries have changed due to removal attacks. Compared to single-point fingerprints, the distinctive verification approach of regarding the whole trajectory as one fingerprint can capture richer decision boundary information, thereby significantly enhancing robustness against removal attacks and reducing the possibility of false positive identification.

### D.5 Time Overheads

As our fingerprinting trajectories requiring iterative optimization, ADV-TRA incurs a higher generation time compared to single-point fingerprinting methods. Table 4 reports the time required to generate 10 trajectories (each containing $40$ samples) and an equivalent set of $400$ single-point fingerprints using the PGD algorithm. It is evident that the time needed to generate trajectories is significantly less than the time required to train a model. Therefore, it is acceptable for the model owner to generate a number of fingerprinting trajectories for future ownership verification. Furthermore, in our subsequent experiments (see Appendix D.2 and D.3), we find that the time overheads can be further reduced (while achieving decent performance) by shortening the trajectory length and decreasing the number of classes spanned by the trajectory. During the ownership verification phase,

Table 4: Time overheads for training the model from scratch, PGD-fingerprinting, and ADV-TRA. Note that we use a pre-trained model for the ImageNet case.

|  | CIFAR-10 | CIFAR-100 | ImageNet |
|---|---|---|---|
| Model Training | $1h37m$ | $2h41m$ | - |
| PGD | $49s$ | $52s$ | $2m55s$ |
| **Ours** | $5m42s$ | $5m53s$ | $27m13s$ |

since querying the black-box model is almost real-time, we do not discuss the time overheads at this stage.

# E   Discussions

**Expansibility.** In our design, the trajectory generation method that records the evolution process of adversarial samples is built upon the classical Basic Iterative Method [29] by incorporating a series of optimization strategies. More recent works towards more undetectable [49, 50] and transferable [51, 52] adversarial samples can also be combined with our method to further enhance the protection performance.

**Universality.** Most existing model watermarking or fingerprinting methods mainly focus on image classification models and cannot be directly extended to other types of DNN models, such as generative models. When facing new model architectures and input-output mappings, extending our approach to other models can be challenging. To the best of our knowledge, our approach is the only method so far that quantifies model decision boundary altering through applying adversarial perturbations at different levels, which may allow it to adapt to other models more readily compared to single-point fingerprinting methods.

**Advanced Removal Attacks.** First, in realistic scenarios, an attacker who attempts to remove fingerprints from a target model via removal attacks may not rely on a single type of attack, but instead employ a combination of several attack methods to achieve better fingerprint removal. In such cases, the reliability of existing fingerprinting schemes may be significantly undermined. Second, our experimental results show that adversarial training poses a greater threat compared to other removal attacks. This is possible because adversarial training effectively improves the robustness of the model, i.e. the ability of the model to resist adversarial samples. That is to say, the model's robustness may instead degrade the effectiveness of fingerprinting methods. In addition to adversarial training, there are still many other techniques that can improve model robustness, such as adding regularization terms during training. Therefore, further research is needed to investigate the impact of these model robustness enhancement techniques on model fingerprinting.

**Limitations.** Unlike traditional single-point fingerprinting methods where each fingerprinting sample corresponds to a single sample, we use a chain of samples (i.e., the trajectory) to jointly fingerprint the decision surface. While our fingerprinting method is able to accurately capture the global and more rich fingerprint features, it inevitably increases the number of queries needed. For example, the total length of a trajectory that crosses $m$ classes, with $2l$ samples in each class, is $m \times 2l$. This also means the number of queries required for a trajectory during the verification phase. In practice, multiple trajectories are often needed to reduce randomness. For DNN models deployed in the cloud as prediction interfaces, too many queries may increase the cost of verification (Of course, model owners can apply for unrestricted/unobstructed model verification from third-party copyright organizations). Therefore, in our design we have take efforts to restrict the needed querying costs by using a fix-length trajectory, instead of the traditional adversarial sample induced trajectory with uncontrollable length. In our future work, we plan to explore the adversarial trajectory generation method using even fewer queries to further improve the querying efficiency.

**Broader Impacts.** In this paper, we propose ADV-TRA for protecting the copyright of deep learning models. Under the black-box deployment of suspect models, it utilizes the progressive adversarial perturbations to detect similarities in the decision boundaries of the models. This new paradigm exhibits stronger robustness and better decision boundary localization capability compared to existing single-point fingerprinting methods, especially when facing removal attacks. Our approach not only provides a new perspective for protecting the IP of DNNs but also promotes the development of

interpretability for black-box models. As a method for protecting model copyrights, this paper is dedicated to safeguarding the rights of legitimate model owners. In summary, this paper drives the development of the copyright community in the AI era, providing security guarantees for the advancement of AI.

# F    Additional Information

## F.1    Experiment Environment

**Hardware Information.** We conduct all experiments on a server equipped with:
- **CPU:** An AMD Ryzen 9 7950X CPU 16-Core@4.50GHz
- **GPU:** 2 NVIDIA GeForce RTX 4090 GPUs each with 24 GB of memory
- **Memory:** 64 GB, DDR4, 5600MHz

**Operating Systems and Environment.**
- **Operating System:** Windows 11 Professional 22H2
- **Environment:** We use Anaconda to build Python environment and Spyder as the compiler. All experiments are implemented based on Pytorch [53].
- **Libraries:** Python=3.8.15, torch=1.12.0, torchvision=0.13.0, numpy=1.23.4, advertorch=0.2.3, scikit-learn=0.20.0

## F.2    PyTorch Code

The code of ADV-TRA is available at: `https://github.com/SPHelixLab/ADV-TRA`.

**Getting Started.** To run this repository, we kindly advise you to install Python 3.8 and PyTorch 1.12 with Anaconda. You can download Anaconda and read the installation instructions on the official website[10].

Create a new virtual environment named "ADV-TRA" based on Python 3.8 and enter this environment:

```
conda create -n ADV_TRA python=3.8
conda activate ADV_TRA
```

Install PyTorch and related packages in torch environment:

```
conda install pytorch==1.12.1
conda install torchvision -c pytorch
pip install numpy 1.23.4 advertorch=0.2.3 scikit-learn 0.20.0
```

**File Structure.** We now introduce the composition of Python files and their implemented functions in this project.

Process the raw dataset and partition it for attacker, defender, and test set:

```
./utils/data_process.py
```

Train the source model:

```
./utils/utils.py
```

Include the design of the model structure involved in this paper:

```
./utils/models.py
```

Generate adversarial trajectories as the fingerprints as well as verify the suspect model with a number of adversarial trajectories:

```
./utils/adv_gen.py
```

The main program for our framework:

```
1        ./utils/main.py
```

Note that the main program contains numerous parameters. You can print the help descriptions for all parameters with the following command:

```
1        python main.py -h
```

